# Stereopsis by a Neural Network Which Learns the Constraints

**Alireza Khotanzad and Ying-Wung Lee**
Image Processing and Analysis Laboratory
Electrical Engineering Department
Southern Methodist University
Dallas, Texas 75275

## Abstract

This paper presents a neural network (NN) approach to the problem of stereopsis. The correspondence problem (finding the correct matches between the pixels of the epipolar lines of the stereo pair from amongst all the possible matches) is posed as a non-iterative many-to-one mapping. A two-layer feed forward NN architecture is developed to learn and code this nonlinear and complex mapping using the back-propagation learning rule and a training set. The important aspect of this technique is that none of the typical constraints such as uniqueness and continuity are explicitly imposed. All the applicable constraints are learned and internally coded by the NN enabling it to be more flexible and more accurate than the existing methods. The approach is successfully tested on several random-dot stereograms. It is shown that the net can generalize its learned mapping to cases outside its training set. Advantages over the Marr-Poggio Algorithm are discussed and it is shown that the NN performance is superior.

## 1 INTRODUCTION

Three-dimensional image processing is an indispensable property for any advanced computer vision system. Depth perception is an integral part of 3-d processing. It involves computation of the relative distances of the points seen in the 2-d images to the imaging device. There are several methods to obtain depth information. A common technique is stereo imaging. It uses two cameras displaced by a known distance to generate two images of the same scene taken from these two different viewpoints. Distances to objects can be computed if corresponding points are identified in both frames. Corresponding points are two image points which correspond to the same object point in the 3-d space as seen by the left and the right cameras, respectively. Thus, solving the so called "correspondence problem"

is the essential stage of depth perception by stereo imaging.

Many computational approaches to the correspondence problem have been studied in the past. An exhaustive review of such techniques is best left to a survey articles by Dhond and Aggarwal (1989). Common to all such techniques is the employment of some constraints to limit computational requirement and also reduce the ambiguity. They usually consist of strict rules that are fixed *a priori* and are based on a rough model of the surface to-be-solved. Unfortunately, psychophysical evidence of human stereopsis suggest that the appropriate constraints are more complex and more flexible to be characterized by simple fixed rules.

In this paper, we suggest a novel approach to the stereo correspondence problem via neural networks (NN). The problem is cast into a mapping framework and subsequently solved by a NN which is especially suited to such tasks. An important aspect of this approach is that the appropriate constraints are automatically learned and generalized by the net resulting in a flexible and more accurate model.

The iterative algorithm developed by Marr and Poggio (1976) for can be regarded as a crude neural network approach with no embedded learning. In fact, the initial stages of the proposed technique follow the same initial steps taken in that algorithm. However, the later stages of the two algorithms are quite distinct with ours involving a learning process and non-iterative operation.

There have been other recent attempts to solve the correspondence problem by neural networks. Among these are O'Toole (1989), Qian and Sejnowski (1988), Sun et al. (1987), and Zhou and Chellappa (1988). These studies use different approaches and topologies from the one used in this paper.

## 2   DESCRIPTION OF THE APPROACH

The proposed approach poses the correspondence problem as a mapping problem and uses a special kind of NN to learn this mapping. The only constraint that is explicitly imposed is the "epipolar" constraint. It states that the match of a point in row m of one of the two images can only be located in row m of the other image. This helps to reduce the computation by restricting the search area.

### 2.1  CORRESPONDENCE PROBLEM AS A MAPPING PROBLEM

The initial phase of the procedure involves casting the correspondence problem as a many to one mapping problem. To explain the method, let us consider a very simple problem involving one row (epipolar line) of a stereo pair. Assume 6 pixel wide rows and take the specific example of [001110] and [111010] as left and right image rows respectively. The task is to find the best possible match between these two strings which in this case is [1110].

The process starts by forming an "initial match matrix". This matrix includes all possible matches between the pixels of the two rows. Fig. 1 illustrates this matrix for the considered example. Each 1 indicates a potential match. However only a few of these matches are correct. Thus, the main task is to distinguish the correct matches which are starred from the false ones.

To distinguish the correct matches from the false ones, Marr and Poggio (1976) imposed two constraints on the correspondences; (1) uniqueness- that there should be a one-to-one correspondence between features in the two eyes, and (2) smoothness - that surfaces should change smoothly in depth. The first constraint means that only one element of the match matrix may have a value of 1 along each horizontal and vertical direction. The second constraint translates into a tendency for the correct matches to spread along the 45° directions. These constraints are implemented through weighted connections between match matrix elements. The uniqueness constraint is modeled by inhibitory (negative) weights along the horizontal/vertical directions. The smoothness constraint gives rise to excitatory (positive) weights along 45° lines. The connections from the rest of elements receive a zero (don't care) weight. Using fixed excitatory and inhibitory constants, they progressively eliminate false correspondences by applying an iterative algorithm.

The described row wise matching does not consider the vertical dependency of pixels in 2-d images. To account for inter-row relationships, the procedure is extended by stacking up the initial match matrices of all the rows to generate a three-dimensional "initial match volume", as shown in Fig. 2. Application of the two mentioned constraints extends the 2-d excitatory region described above to a 45° oriented plane in the volume while the inhibitory region remains on the 2-d plane of the row-wise match. Since depth changes usually happen within a locality, instead of using the complete planes, a subregion of them around each element is selected. Fig. 3 shows an example of such a neighborhood. Note that the considered excitatory region is a circular disc portion of the 45° plane. The choice of the radius size (three in this case) is arbitrary and can be varied. A similar iterative technique is applied to the elements of the initial match volume in order to eliminate incompatible matches and retain the good ones.

There are several serious difficulties with the Marr-Poggio algorithm. First, there is no systematic method for selection of the best values of the excitatory/inhibitory weights. These parameters are usually selected by trial and error. Moreover, a set of weights that works well for one case does not necessarily yield good results for a different pair of images. In addition, utilization of constant weights has no analogy in biological vision systems. Another drawback regards the imposition of the two previously mentioned constraints which are based on assumptions about the form of the underlying scene. However, psychophysical evidence suggests that the stereopsis constraints are more complex and more flexible than can be characterized by simple fixed rules.

The view that we take is that the described process can be posed as a mapping operation from the space of "initial match volume" to the space of "true match volume". Such a transformation can be considered as a one-shot (non-iterative) mapping from the initial matches to the final ones. This is a complex non-linear relationship which is very difficult to model by conventional methods. However, a neural net can learn, and more importantly generalize it.

## 2.2   NEURAL NETWORK ARCHITECTURE

The described mapping is a function of the elements in the initial match volume. This can be expressed as:

$$t(x_1, x_2, x_3) = f\ (i(a, b, c)\ |\ (a, b, c)\ \epsilon\ S)$$

where

$t(x_1, x_2, x_3) =$      state of the node located at coordinate $(x_1, x_2, x_3)$ in the true match volume.

$f =$      the nonlinear mapping function.

$i(a, b, c) =$      state of the node located at coordinate $(a, b, c)$ in the initial match volume.

$S =$      A set of three-dimensional coordinates including $(x_1, x_2, x_3)$ and those of its neighbors in a specified neighborhood.

In such a formulation, if f is known, the task is complete. A NN is capable of learning f through examining a set of examples involving initial matches and their corresponding true matches. The learned function will be coded in a distributive manner as the learned weights of the net.

Note that this approach does not impose any constraints on the solution. No *a priori* excitatory/inhibitory assignments are made. Only a unified concept of a neighboring region, S, which influences the disparity computation is adopted. The influence of the elements in S on the solution is learned by the NN. This means that all the appropriate constraints are automatically learned.

Unlike the Marr-Poggio approach, the NN formulation allows us to consider any shape or size for the neighborhood, S. Although in discussions of next sections we use a Marr-Poggio type neighborhood as shown in Fig. 3, there is no restriction on this. In this work we used this S in order to be able to compare our results with those of Marr-Poggio. In a previous study (Khotanzad & Lee (1990)) we used a standard fully connected multi-layer feed-forward NN to learn f. The main problem with that net is the ad hoc selection of the number of hidden nodes. In this study, we use another layered feed-forward neural net termed "sparsely connected NN with augmented inputs" which does not suffer from this problem. It consists of an input layer, an output layer, and one "hidden layer. The hidden layer nodes and the output node have a Sigmoid non-linearity transfer function. The inputs to this net consist of the state of the considered element in the initial match volume along with states of those in its locality as will be described. The response of the output node is the computed state of the considered element of the initial match volume in the true match volume. The number of hidden nodes are decided based on the shape and size of the selected neighborhood, S , as described in the example to follow. This net is not a fully connected net and each hidden node gets connected to a subset of inputs. Thus the term "sparsely connected" is used.

To illustrate the suggested net, let us use the S of Fig. 3. In this case, each element in the initial match volume gets affected by 24 other elements shown by circles and crosses in the figure. Our suggested network for such an S is shown in Fig. 4. It has 625 inputs, 25 hidden nodes and one output node. Each hidden node is only connected to one set of 25 input nodes. The 625 inputs consist of 25 sets of 25 elements of the initial match volume. Let us denote these sets by $I_1, I_2, \cdots, I_{25}$ respectively. The first set of 25 inputs consists of the state of the element of the initial match volume whose final state is sought along with those of

its 24 neighbors. Let us denote this node and its neighbors by t and $S^t = s_1^t, s_2^t, \cdots, s_{24}^t$ respectively. Then $I_1 = \{t, S^t\}$. The second set is composed of the same type of information for neighbor $s_1^t$. In other words $I_2 = \{s_1^t, S^{s_1^t}\}$. $I_3, \cdots, I_{25}$ are made similarly. So in general

$$I_j = \{s_j^t, S^{s_j^t}\}, \quad j = 2, 3,..., 25.$$

Note that there is a good degree of overlap among these 625 inputs. However, these redundant inputs are processed separately in the hidden layer as explained later. Due to the structure of this input, it is referred to as "augmented input".

The hidden layer consists of 25 nodes, each of which is connected to only one of the 25 sets of inputs through weights to be learned. Thus, each node of the hidden layer processes the result of evolution of one of the 25 input sets. The effects of processing these 25 evolved sets would then be integrated at the single output node through the connection weights between the hidden nodes and the output node. The output node then computes the corresponding final state of the considered initial match element.

Training this net is equivalent to finding proper weights for all of its connections as well as thresholds associated with the nodes. This is carried out by the back-propagation learning algorithm (Rumelhart et. al (1986)). Again note that all the weights used in this scheme are unknown and need to be computed through the learning procedure with the training set. Thus, the concept of *a priori* excitatory and inhibitory labeling is not used.

## 3   EXPERIMENTAL STUDY

The performance of the proposed neural network approach is tested on several random-dot stereograms. A random dot stereogram consists of a pair of similar structural images filled with randomly generated black and white dots, with some regions of one of the images shifted to either left or right relative to the other image. When viewed through a stereoscope, a human can perceive the shifted structures as either floating upward or downward according to their relative disparities. Stereograms with 50% density (i.e. half black, half white) are used.

Six 32×32 stereograms with varying disparities are used to teach the network. The actual disparity maps (floating surfaces) of these are shown in Fig. 5. Each stereogram contains three different depth levels (disparity regions) represented by different gray levels. Therefore, six three-dimensional initial match volumes and their six corresponding true match volumes comprise the training set for the NN. Each initial match volume and its corresponding true match volume contain $32^3$ input-output pairs. Since six stereograms are considered, a total of $6 \times 32^3$ input-output pairs are available for training.

The performance of the trained net is tested on several random-dot stereograms. Fig. 5 shows the results for the same data the net is trained with. In addition the performance was tested on other stereograms that are different from the training set. The considered differences include: the shape of the disparity regions, size of the image, disparity levels, and addition of noise to one image of the pair. These cases are not presented here due to space limitation. We can report that all of them yielded very good results.

In Fig. 5, the results obtained using the Marr-Poggio algorithm are also shown for comparison. Even though it was tried to find the best feed backs for Marr-Poggio through trial and error, the NN outperformed it in all cases in terms of number of error pixels in the resulting disparity map.

# 4    CONCLUSION

In this paper, a neural network approach to the problem of stereopsis was discussed. A multilayer feed-forward net was developed to learn the mapping that retains the correct matches between the pixels of the epipolar lines of the stereo pair from amongst all the possible matches. The only constraint that is explicitly imposed is the "epipolar" constraint. All the other appropriate constraints are learned by example and coded in the nets in a distributed fashion. The net learns by examples of stereo pairs and their corresponding depth maps using the back-propagation learning rule. Performance was tested on several random-dot stereograms and it was shown that the learning is generalized to cases outside the training. The net performance was also found to be superior to Marr-Poggio algorithm.

**Acknowledgements**

This work was supported in part by DARPA under Grant MDA-903-86-C-0182

**References**

Dhond, U. R. & Aggarwal, J. K. (1989), "Structure from stereo - A review," *IEEE Trans. SMC*, vol. 19, pp. 1489-1510.

Drumheller, M. & Poggio, T. (1986), "On parallel stereo," *Proc. IEEE Intl. Conf. on Robotics and Automation*, vol. 3, pp. 1439-1448.

Khotanzad, A. & Lee, Y. W. (1990), "Depth Perception by a Neural Network," *IEEE Midcon/90 Conf. Record*, Dallas, Texas, pp. 424-427, Sept. 11-13.

Marr, D. & Poggio, T. (1976), "Cooperative computation of stereo disparity," Science, 194, pp. 238-287.

O'Toole, A. J. (1989), "Structure from stereo by associative learning of the constraints," *Perception*, 18, pp. 767-782.

Poggio, T. (1984), "Vision by man and machine," *Scientific American*, vol. 250, pp. 106-116, April.

Qiang, N. & Sejnowski, T. J. (1988), "Learning to solve random-dot stereograms of dense and transparent surfaces with recurrent backpropagation," in Touretzky & Sejnowski (Eds.), Proceedings of the 1988 Connectionist Models, pp. 435-444, Morgan Kaufmann Publishers.

Rumelhart, D. E., Hinton G. E., and Williams R. J. (1986), "Learning internal representations by error propagation," in D.E. Rumelhart & J.L. McClelland (Eds.), Parallel Distributed Processing: Explorations in the Microstructure of Cognition. vol. 1: Foundations, MIT Press.

Sun, G. Z., Chen, H. H., Lee, Y. C. (1987), "Learning stereopsis with neural networks," *Proc. IEEE First Intl. Conf. on Neural Networks*, San Diego, CA, pp. 345-355, June.

Zhou, Y. T. and Chellappa, R. (1988), "Stereo matching using a neural network," *Proc. IEEE International Conf. Acoustics, Speech, and Signal Processing, ICASSP-88*, New York, pp. 940-943, April 11-14.

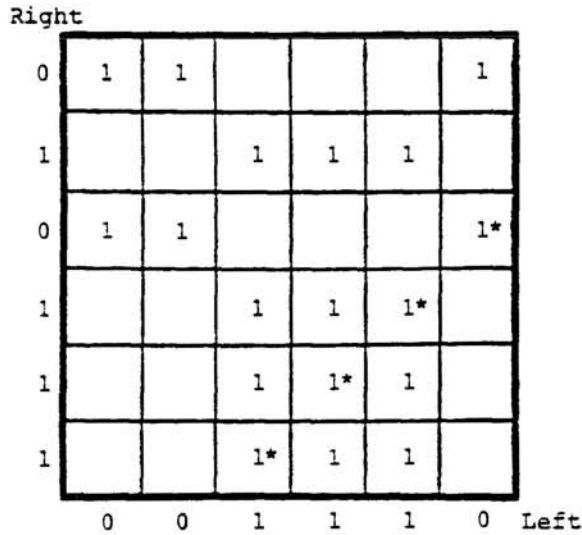

Figure 1: The initial match matrix for the considered example. 1 represents a match. Correct matches are starred.

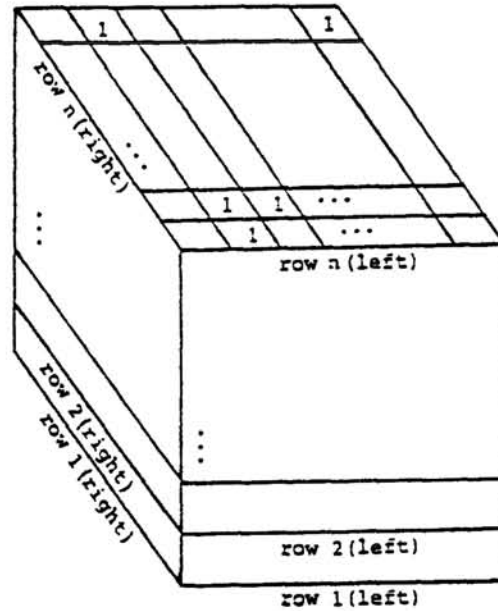

Figure 2: Schematic of the initial match volume constructed by stacking up row match matrices.

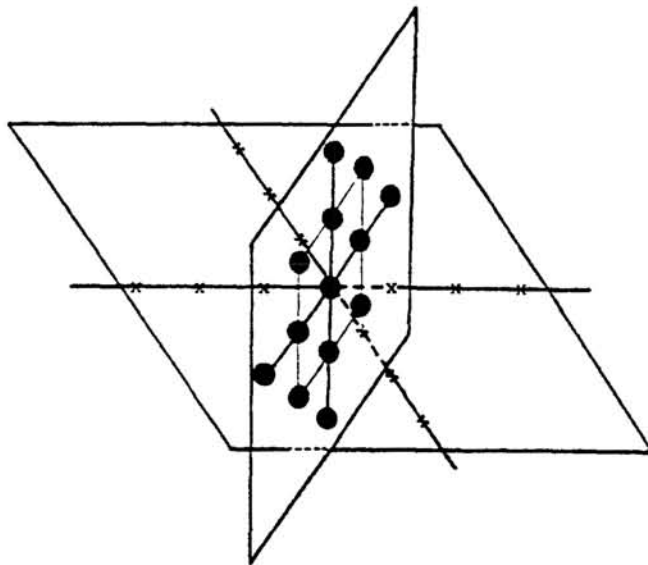

Figure 3: The neighborhood structure considered in the initial match volume. If used with Marr-Poggio, circles and crosses represent excitatory and inhibitory neighbors respectively.

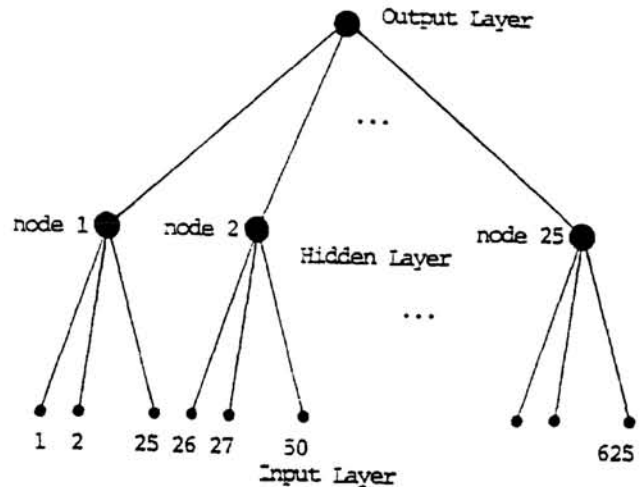

Figure 4: The sparsely connected NN with augmented inputs when the neighborhood of Fig. 3 is used.

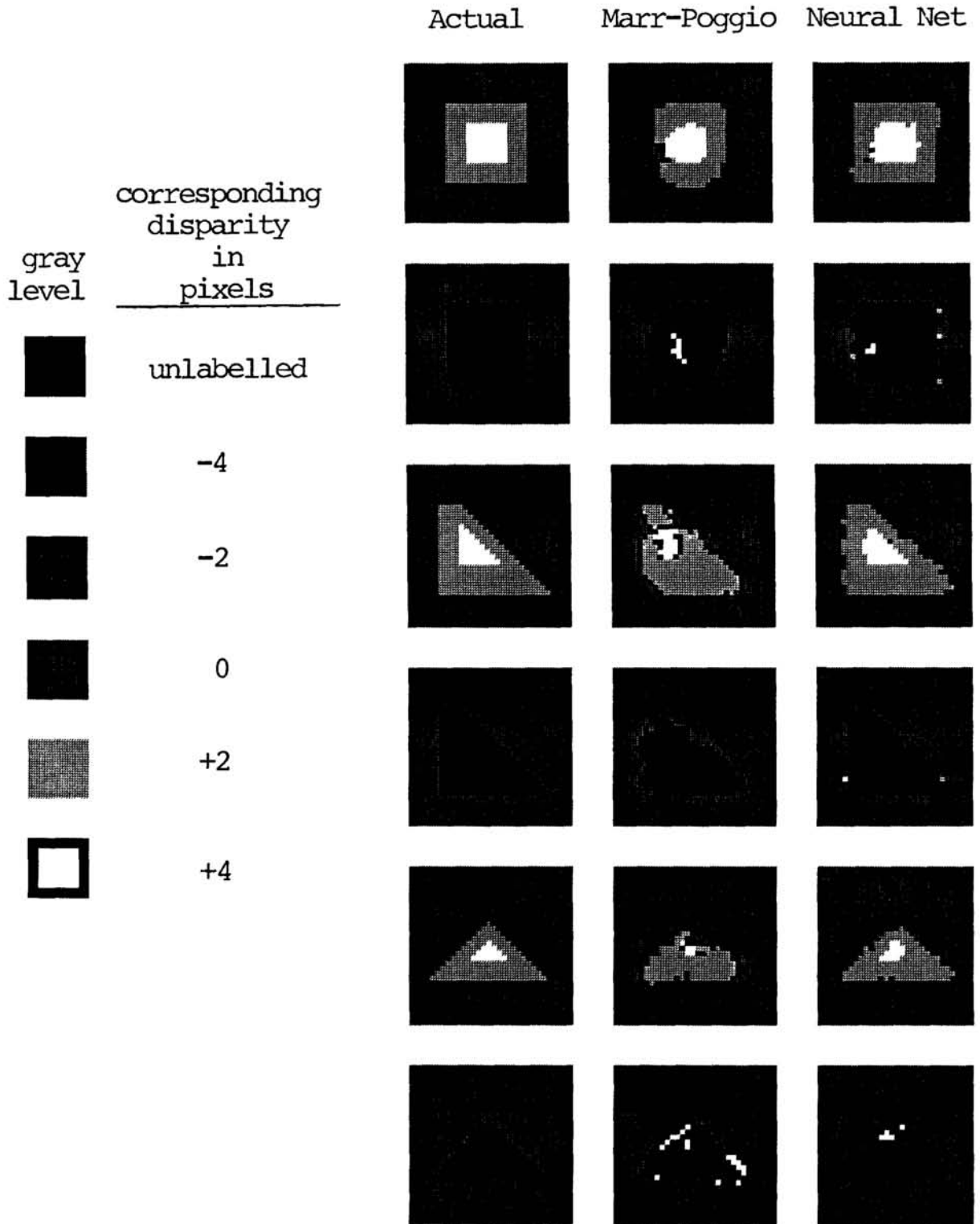

Figure 5: The results of disparity computation for six random-dot stereograms which are used to train the NN. The Marr-Poggio results are also shown.